# Half-Lives of EigenFlows for Spectral Clustering

**Chakra Chennubhotla & Allan D. Jepson**
Department of Computer Science, University of Toronto, Canada M5S 3H5
{chakra,jepson}@cs.toronto.edu

## Abstract

Using a Markov chain perspective of spectral clustering we present an algorithm to automatically find the number of stable clusters in a dataset. The Markov chain's behaviour is characterized by the spectral properties of the matrix of transition probabilities, from which we derive eigenflows along with their halflives. An eigenflow describes the flow of probability mass due to the Markov chain, and it is characterized by its eigenvalue, or equivalently, by the halflife of its decay as the Markov chain is iterated. A ideal stable cluster is one with zero eigenflow and infinite half-life. The key insight in this paper is that bottlenecks between weakly coupled clusters can be identified by computing the sensitivity of the eigenflow's halflife to variations in the edge weights. We propose a novel EIGENCUTS algorithm to perform clustering that removes these identified bottlenecks in an iterative fashion.

## 1 Introduction

We consider partitioning a weighted undirected graph— corresponding to a given dataset— into a set of discrete clusters. Ideally, the vertices (i.e. datapoints) in each cluster should be connected with high-affinity edges, while different clusters are either not connected or are connnected only by a few edges with low affinity. The practical problem is to identify these tightly coupled clusters, and cut the inter-cluster edges.

Many techniques have been proposed for this problem, with some recent success being obtained through the use of spectral methods (see, for example, [2, 4, 5, 11, 12]). Here we use the random walk formulation of [4], where the edge weights are used to construct a Markov transition probability matrix, $M$. This matrix $M$ defines a random walk on the graph to be partitioned. The eigenvalues and eigenvectors of $M$ provide the basis for deciding on a particular segmentation. In particular, it has been shown that for $K$ weakly coupled clusters, the leading $K$ eigenvectors of $M$ will be roughly piecewise constant [4, 13, 5]. This result motivates many of the current spectral clustering algorithms. For example in [5], the number of clusters $K$ must be known *a priori*, and the $K$-means algorithm is used on the $K$ leading eigenvectors of $M$ in an attempt to identify the appropriate piecewise constant regions.

In this paper we investigate the form of the leading eigenvectors of the Markov matrix $M$. Using some simple image segmentation examples we confirm that the leading eigenvectors of $M$ are roughly piecewise constant for problems with well separated clusters. However, we observe that for several segmentation problems that we might wish to solve, the coupling between the clusters is significantly stronger and, as a result, the piecewise constant approximation breaks down.

Unlike the piecewise constant approximation, a perfectly general view is that the eigenvectors of $M$ determine particular flows of probability along the edges in the graph. We refer to these as eigenflows since they are characterized by their associated eigenvalue $\lambda$, which specifies the flow's overall rate of decay. Instead of measuring the decay rate in terms of the eigenvalue $\lambda$, we find it more convenient to use the flow's halflife $\beta$, which is simply defined by $|\lambda|^{\beta} = 1/2$. Here $\beta$ is the number of Markov chain steps needed to reduce the particular eigenflow to half its initial value. Note that as $\lambda$ approaches 1 the half-life approaches infinity.

From the perspective of eigenflows, a graph representing a set of weakly coupled clusters produces eigenflows between the various clusters which decay with long halflives. In contrast, the eigenflows within each cluster decay much more rapidly. In order to identify clusters we therefore consider the eigenflows with long halflives. Given such a slowly decaying eigenflow, we identify particular bottleneck regions in the graph which critically restrict the flow (cf. [12]). To identify these bottlenecks we propose computing the sensitivity of the flow's halflife with respect to perturbations in the edge weights.

We implement a simple spectral graph partitioning algorithm which is based on these ideas. We first compute the eigenvectors for the Markov transition matrix, and select those with long halflives. For each such eigenvector, we identify bottlenecks by computing the sensitivity of the flow's halflife with respect to perturbations in the edge weights. In the current algorithm, we simply select one of these eigenvectors in which a bottleneck has been identified, and cut edges within the bottleneck. The algorithm recomputes the eigenvectors and eigenvalues for the modified graph, and continues this iterative process until no further edges are cut.

## 2 From Affinities to Markov Chains

Following the formulation in [4], we consider an undirected graph $G$ with vertices $v_i$, for $i = 1, \ldots, N$, and edges $e_{i,j}$ with non-negative weights $a_{i,j}$. Here the weight $a_{i,j}$ represents the affinity of vertices $v_i$ and $v_j$. The edge affinities are assumed to be symmetric, that is, $a_{i,j} = a_{j,i}$. A Markov chain is defined using these affinities by setting the transition probability $m_{i,j}$ from vertex $v_j$ to vertex $v_i$ to be proportional to the edge affinity, $a_{i,j}$. That is, $m_{i,j} = d_j^{-1} a_{i,j}$ where $d_j = \sum_{i=1}^n a_{i,j}$ gives the normalizing factor which ensures $\sum_{i=1}^n m_{i,j} = 1$. In matrix notation, the affinities are represented by a symmetric $n \times n$ matrix $A$, with elements $a_{i,j}$, and the transition probability matrix $M = (m_{i,j})$ is given by

$$M = AD^{-1}, \quad D = \text{diag}(d_1, \ldots, d_n). \tag{1}$$

Notice that the $n \times n$ matrix $M$ is not in general symmetric.

This transition probability matrix $M$ defines the random walk of a particle on the graph $G$. Suppose the initial probability of the particle being at vertex $v_j$ is $p_j^0$, for $j = 1, \ldots, n$. Then, the probability of the particle being initially at vertex $v_j$ and taking edge $e_{i,j}$ is $m_{i,j} p_j^0$. In matrix notation, the probability of the particle ending up any of the vertices $\vec{v} = (v_1, v_2, \cdots, v_n)$ after one step is given by the distribution $\vec{p}^1 = M\vec{p}^0$, where $\vec{p}^k = (p_1^k, \ldots, p_n^k)$.

For analysis it is convenient to consider the matrix $L = D^{-1/2} M D^{1/2}$, which is similar to $M$ (where $D$ is as given in Eq. (1)). The matrix $L$ therefore has the same spectrum as $M$ and any eigenvector $\vec{u}$ of $L$ must correspond to an eigenvector $D^{1/2}\vec{u}$ of $M$ with the same eigenvalue. Note that $L = D^{-1/2} M D^{1/2} = D^{-1/2} A D^{-1} D^{1/2} = D^{-1/2} A D^{-1/2}$, and therefore $L$ is a symmetric $n \times n$ matrix since $A$ is symmetric while $D$ is diagonal.

The advantage of considering the matrix $L$ over $M$ is that the symmetric eigenvalue problem is more stable to small perturbations, and is computationally much more tractable. Since the matrix $L$ is symmetric, it has an orthogonal decomposition of the form:

$$L = U \Lambda U^T, \tag{2}$$

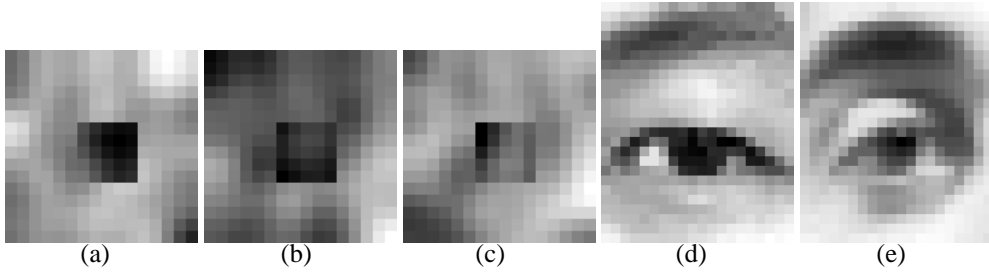

| (a) | (b) | (c) | (d) | (e) |

Figure 1: (a-c) Three random images each having an occluder in front of a textured background. (d-e) A pair of eye images.

where $U = [\vec{u}_1, \vec{u}_2, \cdots, \vec{u}_n]$ are the eigenvectors and $\Lambda$ is a diagonal matrix of eigenvalues $[\lambda_1, \lambda_2, \cdots, \lambda_n]$ sorted in decreasing order. While the eigenvectors have unit length, $\|\vec{u}_k\| = 1$, the eigenvalues are real and have an absolute value bounded by 1, $|\lambda_k| \leq 1$.

The eigenvector representation provides a simple way to capture the Markovian relaxation process [12]. For example, consider propagating the Markov chain for $\beta$ iterations. The transition matrix after $\beta$ iterations, namely $M^\beta$, can be represented as:

$$M^\beta = D^{1/2} U \Lambda^\beta U^T D^{-1/2}. \tag{3}$$

Therefore the probability distribution for the particle being at vertex $v_i$ after $\beta$ steps of the random walk, given that the initial probability distribution was $\vec{p}^0$, is $\vec{p}^\beta = M^\beta \vec{p}^0 = D^{1/2} U \Lambda^\beta \vec{r}^0$, where $\vec{r}^0 = U^T D^{-1/2} \vec{p}^0$ provides the expansion coefficients of the initial distribution $\vec{p}^0$ in terms of the eigenvectors of $L$. As $\beta \to \infty$, the Markov chain approaches the stationary distribution $\vec{\pi}$, $M^\infty = \vec{\pi} \mathbf{1}^T$. Assuming the graph $G$ is connected with edges having non-zero weights, it is convenient to interpret the Markovian relaxation process as perturbations to the stationary distribution, $\vec{p}^\beta = \vec{\pi} + \sum_{j=2}^n r_j \lambda_j^\beta \vec{q}_j$, where $\lambda_1 = 1$ is associated with the stationary distribution $\vec{\pi}$ and $\vec{q}_j = D^{1/2} \vec{u}_j$.

## 3 EigenFlows

Let $\vec{p}^0$ be an initial probability distribution for a random particle to be at the vertices of the graph $G$. By the definition of the Markov chain, recall that the probability of making the transition from vertex $v_j$ to $v_i$ is the probability of starting in vertex $v_j$, times the conditional probability of taking edge $e_{i,j}$ given that the particle is at vertex $v_j$, namely $m_{i,j} p_j^0$. Similarly, the probability of making the transition in the reverse direction is $m_{j,i} p_i^0$. The net flow of probability mass along edge $e_{i,j}$ from $v_j$ to $v_i$ is therefore the difference $m_{i,j} p_j^0 - m_{j,i} p_i^0$. It then follows that the net flow of probability mass from vertex $v_j$ to $v_i$ is given by $F_{i,j}(\vec{p}^0)$, where $F_{i,j}(\vec{p}^0)$ is the $(i,j)$-element of the $n \times n$ matrix

$$F(\vec{p}^0) = M \mathrm{diag}(\vec{p}^0) - \mathrm{diag}(\vec{p}^0) M^T. \tag{4}$$

Notice that $F = E - E^T$ for $E = M \mathrm{diag}(\vec{p}^0)$, and therefore $F$ is antisymmetric (i.e. $F^T = -F$). This expresses the fact that the flow $F_{j,i}$ from $v_i$ to $v_j$ is just the opposite sign of the flow in the reverse direction. Furthermore, it can be shown that $F(\pi) = 0$ for any stationary distribution $\pi$. Therefore, the flow is caused by the eigenvectors $\vec{q}_j$ with $\lambda_j \neq 1$, and hence we analyze the rate of decay of these eigenflows $F(\vec{q}_j)$.

For illustration purposes we begin by considering an ensemble of random test images formed from two independent samples of 2D Gaussian filtered white noise (see Fig. 1a-c). One sample is used to form the $16 \times 16$ background image, and a cropped $5 \times 5$ fragment of second sample is used for the foreground region. A small constant bias is added to the foreground region.

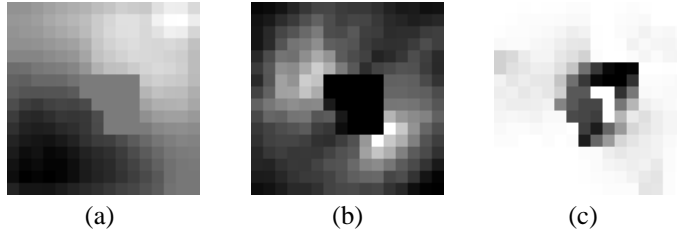

Figure 2: (a) Eigenmode (b) corresponding eigenflow (c) gray value at each pixel corresponds to the maximum of the absolute sensitivities of all the weights on edges connected to a pixel (not including itself). Dark pixels indicate high absolute sensitivities.

A graph clustering problem is formed where each pixel in a test image is associated with a vertex of the graph $G$. The edges in $G$ are defined by the standard 8-neighbourhood of each pixel (with pixels at the edges and corners of the image only having 5 and 3 neighbours, respectively). The edge weight between neighbouring vertices $v_j$ and $v_i$ is given by the affinity $a_{i,j} = \exp\{-(I(\vec{x}_i) - I(\vec{x}_j))^2/(2\sigma^2)\}$, where $I(\vec{x}_k)$ is the test image brightness at pixel $\vec{x}_k$ and $\sigma$ is a grey-level standard deviation. We use $\sigma = \rho g$, where $g$ is the median absolute difference of gray levels between all neighbouring pixels and $\rho = 1.5$.

This generative process provides an ensemble of clustering problems which we feel are representative of the structure of typical image segmentation problems. In particular, due to the smooth variation in gray-levels, there is some variability in the affinities within both foreground and background regions. Moreover, due to the use of independent samples for the two regions, there is often a significant step in gray-level across the boundary between the two regions. Finally, due to the small bias used, there is also a significant chance for pixels on opposite sides of the boundary to have similar gray-levels, and thus high affinities. This latter property ensures that there are some edges with significant weights between the two clusters in the graph associated with the foreground and background pixels.
In Figure 2 we plot one eigenvector, $\vec{q}_k$, of the matrix $M$ along with its eigenflow, $F(\vec{q}^k)$. Notice that the displayed eigenmode is not in general piecewise constant. Rather, the eigenvector is more like vibrational mode of a non-uniform membrane (in fact, they can be modeled in precisely that way). Also, for all but the stationary distribution, there is a significant net flow between neighbours, especially in regions where the magnitude of the spatial gradient of the eigenmode is larger.

## 4   Perturbation Analysis of EigenFlows

As discussed in the introduction, we seek to identify bottlenecks in the eigenflows associated with long halflives. This notion of identifying bottlenecks is similar to the well-known max-flow, min-cut theorem. In particular, for a graph whose edge weights represent maximum flow capacities between pairs of vertices, instead of the current conditional transition probabilities, the bottleneck edges can be identified as precisely those edges across which the maximum flow is equal to their maximum capacity. However, in the Markov framework, the flow of probability across an edge is only maximal in the extreme cases for which the initial probability of being at one of the edge's endpoints is equal to one, and zero at the other endpoint. Thus the max-flow criterion is not directly applicable here.

Instead, we show that the desired bottleneck edges can be conveniently identified by considering the sensitivity of the flow's halflife to perturbations of the edge weights (see Fig. 2c). Intuitively, this sensitivity arises because the flow across a bottleneck will have fewer alternative routes to take and therefore will be particularly sensitive to changes in the edge weights within the bottleneck. In comparison, the flow between two vertices in a strongly coupled cluster will have many alternative routes and therefore will not be particularly

sensitive on the precise weight of any single edge.

In order to pick out larger halflives, we will use one parameter, $\beta_0$, which is a rough estimate of the smallest halflife that one wishes to consider. Since we are interested in perturbations which significantly change the current halflife of a mode, we choose to use a logarithmic scale in halflife. A simple choice for a function which combines these two effects is $S(\beta) = \log(\beta + \beta_0)$, where $\beta$ the halflife of the current eigenmode.

Suppose we have an eigenvector $\vec{u}$ of $L$, with eigenvalue $\lambda$. This eigenvector decays with a halflife of $\beta = -\log(2)/\log(|\lambda|)$. Consider the effect on $S(\beta)$ of perturbing the affinity $a_{i,j}$ for the $(i, j)$-edge, to $a_{i,j} + \alpha_{i,j}$. In particular, we show in the Appendix that the derivative of $S(\beta(\alpha_{i,j}))$ with respect to $\alpha_{i,j}$, evaluated at $\alpha_{i,j} = 0$, satisfies

$$\frac{d\log(\beta + \beta_0)}{d\alpha_{i,j}} = \frac{\log(2)}{\lambda \log(\lambda) \log(\lambda^{\beta_0}/2)} \left[ -\left(\frac{u_i}{\sqrt{d_i}} - \frac{u_j}{\sqrt{d_j}}\right)^2 + (1-\lambda)\left(\frac{u_i^2}{d_i} + \frac{u_j^2}{d_j}\right) \right] \quad (5)$$

Here $(u_i, u_j)$ are the $(i, j)$ elements of eigenvector $\vec{u}$ and $(d_i, d_j)$ are degrees of nodes $(i, j)$ (Eq.1). In Figure 2, for a given eigenvector and its flow, we plot the *maximum* of absolute sensitivities of all the weights on edges connected to a pixel (not including itself). Note that the sensitivities are large in the bottlenecks at the border of the foreground and background.

## 5 EIGENCUTS: A Basic Clustering Algorithm
We select a simple clustering algorithm to test our proposal of using the derivative of the eigenmode's halflife for identifying bottleneck edges. Given a value of $\beta_0$, which is roughly the minimum halflife to consider for any eigenmode, we iterate the following:

1. Form the symmetric $n \times n$ affinity matrix $A$, and initialize $A^c = A$.
2. Set $D_{i,i} = \sum_{j=1}^{n} A_{i,j}^c$, and set a scale factor $\delta$ to be the median of $D_{i,i}$ for $i = 1, \ldots, n$. Form the symmetric matrix $L^c = D^{-1/2} A^c D^{-1/2}$.
3. Compute eigenvectors $[\vec{u}_1, \vec{u}_2, \cdots, \vec{u}_n]$ of $L^c$, with eigenvalues $|\lambda_1| \geq |\lambda_2| \ldots \geq |\lambda_n|$.
4. For each eigenvector $\vec{u}_k$ of $L^c$ with halflife $\beta_k > \epsilon\beta_0$, compute the halflife sensitivities, $S_{i,j}^k = \frac{d\log(\beta_k + \beta_0)}{d\alpha_{i,j}}$ for each edge in the graph. Here we use $\epsilon = 1/4$.
5. Do non-maximal suppression within each of the computed sensitivities. That is, suppress the sensitivity $S_{i,j}^k$ if there is a strictly more negative value $S_{m,j}^k$ or $S_{i,n}^k$ for some vertex $v_m$ in the neighbourhood of $v_j$, or some $v_n$ in the neighbourhood of $v_i$.
6. Compute the sum $T^k$ of $-S_{i,j}^k a_{i,j}^c$ over all non-suppressed edges $(i, j)$ for which $S_{i,j}^k < \tau/\delta$. We use $\tau = -0.1$.
7. Select the eigenmode $\vec{u}_{k*}$ for which $T_{k*}$ is maximal.
8. Cut all edges $(i, j)$ in $A^c$ (i.e. set their affinities to 0) for which $S_{i,j}^{k*} < \tau/\delta$ and for which this sensitivity was not suppressed during non-maximal suppression.
9. If any new edges have been cut, go to 2. Otherwise stop.

Here steps $1 - 3$ are as described previously, other than computing the scaling constant $\delta$, which is used in step 6 to provide a scale invariant threshold on the computed sensitivities. In step 4 we only consider eigenmodes with halflives larger than $\epsilon\beta_0$, with $\epsilon = 1/4$ because this typically eliminates the need to compute the sensitivities for many modes with tiny values of $\beta_k$ and, because of the $\beta_0$ term in $S(\beta)$, it is very rare for eigenvectors with halflives smaller than $\epsilon\beta_0$ to produce any sensitivity less than $\tau$.

In step 5 we perform a non-maximal suppression on the sensitivities for the $k^{th}$ eigenvector. We have observed that at strong borders the computed sensitivities can be less than $\tau$ in a band along the border few pixels thick. This non-maximal suppression allows us to thin this region. Otherwise, many small isolated fragments can be produced in the neighbourhood of such strong borders.

In step 6 we wish to select one particular eigenmode to base the edge cutting on at this iteration. The reason for not considering all the modes simultaneously is that we have found the locations of the cuts can vary by a few pixels for different modes. If nearby edges are cut as a result of different eigenmodes, then small isolated fragments can result in the final clustering. Therefore we wish to select just one eigenmode to base cuts on each iteration. The particular eigenmode selected can, of course, vary from one iteration to the next.

The selection strategy in step 6 above picks out the mode which produces the largest linearized increment in $S(\beta_k) = \log(\beta_k + \beta_0)$. That is, we compute $\Delta S_k = \sum_{i,j} \frac{dS(\beta_k)}{da_{i,j}} \Delta a_{i,j}^c$, where $\Delta a_{i,j}^c = -a_{i,j}^c$ is the change of affinities for any edge left to be cut, and $\Delta a_{i,j}^c = 0$ otherwise. Other techniques for selecting a particular mode were also tried, and they all produced similar results.

This iterative cutting process must eventually terminate since, except for the last iteration, edges are cut each iteration and any cut edges are never uncut. When the process does terminate, the selected succession of cuts provides a modified affinity matrix $A^c$ which has well separated clusters. For the final clustering result, we can use either a connected components algorithm or the $K$-means algorithm of [5] with $K$ set to the number of modes having large halflives.

## 6  Experiments

We compare the quality of EIGENCUTS with two other methods: a $K$-means based spectral clustering algorithm of [5] and an efficient segmentation algorithm proposed in [1] based on a pairwise region comparison function. Our strategy was to select thresholds that are likely to generate a small number of stable partitions. We then varied these thresholds to test the quality of partitions. To allow for comparison with $K$-means, we needed to determine the number of clusters $K$ a priori. We therefore set $K$ to be the same as the number of clusters that EIGENCUTS generated. The cluster centers were initialized to be as orthogonal as possible [5].

The first two rows in Fig. 3 show results using EIGENCUTS. A crucial observation with EIGENCUTS is that, although the number of clusters changed slightly with a change in $\beta_0$, the regions they defined were qualitatively preserved across the thresholds and corresponded to a naive observer's intuitive segmentation of the image. Notice in the random images the occluder is found as a cluster clearly separated from the background. The performance on the eye images is also interesting in that the largely uniform regions around the center of the eye remain as part of one cluster.

In comparison, both the $K$-means algorithm and the image segmentation algorithm of [1] (rows 3-6 in Fig. 3) show a tendency to divide uniform regions and give partitions that are neither stable nor intuitive, despite multiple restarts.

## 7  Discussion

We have demonstrated that the common piecewise constant approximation to eigenvectors arising in spectral clustering problems limits the applicability of previous methods to situations in which the clusters are only relatively weakly coupled. We have proposed a new edge cutting criterion which avoids this piecewise constant approximation. Bottleneck edges between distinct clusters are identified through the observed sensitivity of an eigenflow's halflife on changes in the edges' affinity weights. The basic algorithm we propose is computationally demanding in that the eigenvectors of the Markov matrix must be recomputed after each iteration of edge cutting. However, the point of this algorithm is to simply demonstrate the partitioning that can be achieved through the computation of the sensitivity of eigenflow halflives to changes in edge weights. More efficient updates of the eigenvalue computation, taking advantage of low-rank changes in the matrix $L^c$ from one iteration to the next, or a multi-scale technique, are important areas for further study.

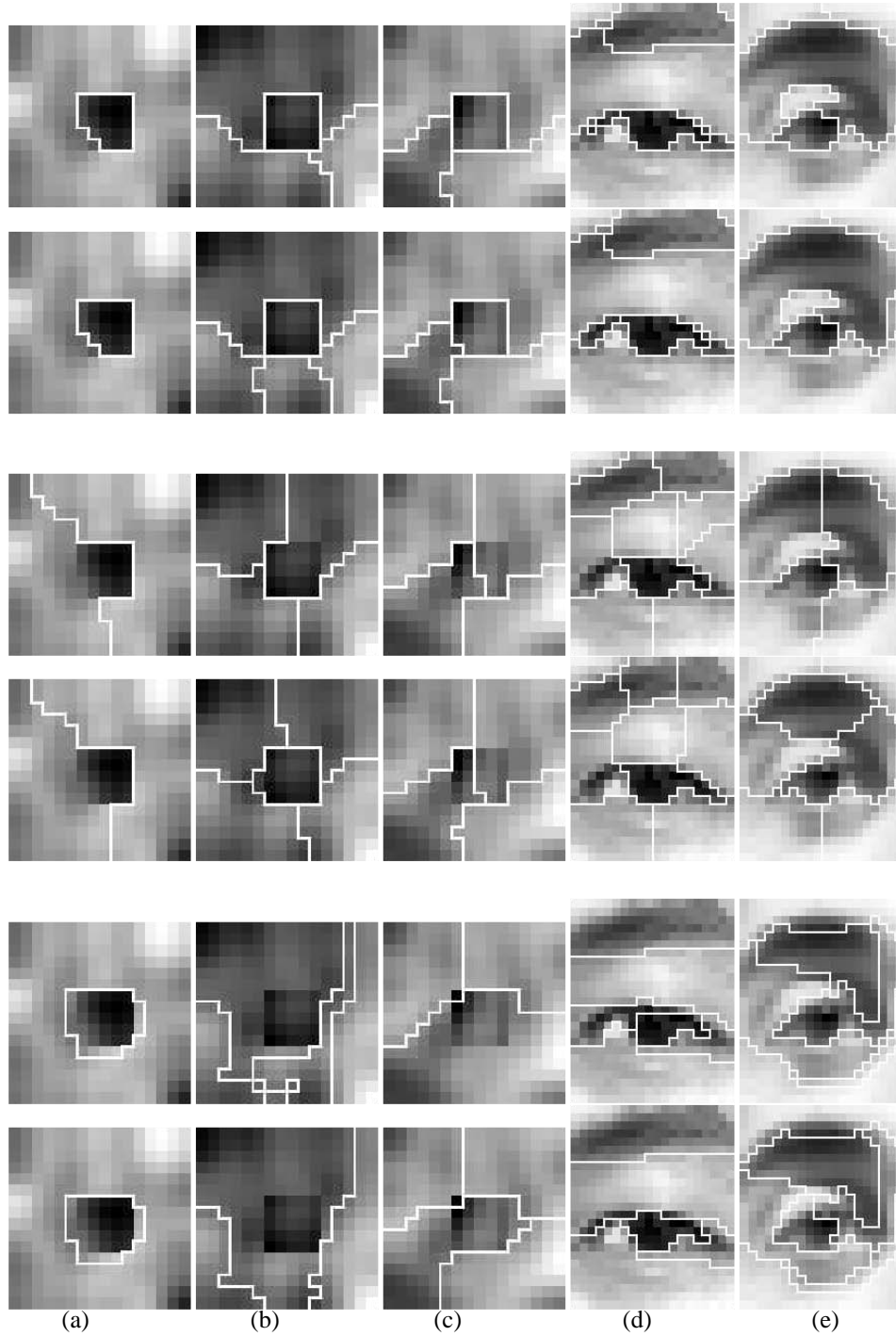

(a)           (b)           (c)           (d)           (e)

Figure 3: Each column refers to a different image in the dataset shown in Fig. 1. Pairs of rows correspond to results from applying: EIGENCUTS with $\rho = 1.5, \epsilon = 0.25, \tau = -0.1$ and $\beta_0 = 80, 60$ (Rows 1&2), $K$-Means spectral clustering where $K$, the number of clusters, is determined by the results of EIGENCUTS (Rows 3&4) and Falsenszwalb & Huttenlocher $\tau = 70, 50$ (Rows 5&6).

## Acknowledgements

We have benefited from discussions with Sven Dickinson, Sam Roweis, Sageev Oore and Francisco Estrada.

## References

[1] P. Felzenszalb and D. Huttenlocher Efficiently Computing a Good Segmentation *Internation Journal on Computer Vision*, 1999.

[2] R. Kannan, S. Vempala and A. Vetta On clusterings–good, bad and spectral. *Proc. 41st Annual Symposium on Foundations of Computer Science* , 2000.

[3] J. R. Kemeny and J. L. Snell Finite Markov Chains. Van Nostrand, New York, 1960.

[4] M. Meila and J. Shi A random walks view of spectral segmentation. *Proc. International Workshop on AI and Statistics* , 2001.

[5] A. Ng, M. Jordan and Y. Weiss On Spectral Clustering: analysis and an algorithm *NIPS*, 2001.

[6] A. Ng, A. Zheng, and M. Jordan Stable algorithms for link analysis. *Proc. 24th Intl. ACM SIGIR Conference*, 2001.

[7] A. Ng, A. Zheng, and M. Jordan Link analysis, eigenvectors and stability. *Proc. 17th Intl. IJCAI*, 2001.

[8] P. Perona and W. Freeman A factorization approach to grouping. *European Conference on Computer Vision*, 1998.

[9] A. Pothen Graph partitioning algorithms with applications to scientific computing. Parallel Numerical Algorithms, D. E. Keyes et al (eds.), Kluwer Academic Press, 1996.

[10] G. L. Scott and H. C. Longuet-Higgins Feature grouping by relocalization of eigenvectors of the proximity matrix. *Proc. British Machine Vision Conference*, pg. 103-108, 1990.

[11] J. Shi and J. Malik. Normalized cuts and image segmentation. *IEEE Transaction on Pattern Analysis and Machine Intelligence* , 2000.

[12] N. Tishby and N. Slonim Data clustering by Markovian Relaxation and the Information Bottleneck Method. *NIPS*, v 13, MIT Press, 2001.

[13] Y. Weiss Segmentation using eigenvectors: a unifying view. *International Conference on Computer Vision*, 1999.

## Appendix

We compute the derivative of the log of half-life $\beta$ of an eigenvalue $\lambda$ with respect to an element $a_{ij}$ of the affinity matrix $A$. Half-life is defined as the power to which $\lambda$ must be raised to reduce the eigenvalue to half, i.e., $\lambda^{\beta} = 1/2$. What we are interested is in seeing significant changes in those half-lives $\beta$ which are relatively large compared to some minimum half-life $\beta_0$. So eigenvectors with half-lives smaller than $\beta_0$ are effectively ignored. It is easy to show that,

$$\frac{d\log(\beta + \beta_0)}{da_{ij}} = \frac{\log(2)}{\lambda \log(\lambda)\log(\lambda^{\beta_0}/2)}\frac{d\lambda}{da_{ij}}, \quad \text{and} \quad \frac{d\lambda}{da_{ij}} = \vec{u}^T \frac{dL}{da_{ij}}\vec{u}. \tag{6}$$

Let $\vec{u}$ be the corresponding eigenvector such that $L\vec{u} = \lambda\vec{u}$, where $L$ is the modified affinity matrix (Sec 2). As $L = D^{-1/2}AD^{-1/2}$, we can write for all $i \neq j$:

$$\frac{dL}{da_{ij}} = D^{-1/2}\big[Z_{ij} + Z_{ji}\big]D^{-1/2} - PAD^{-1/2} - D^{-1/2}AP, \tag{7}$$

where $Z_{gh}$ is a matrix of all zeros except for a value of 1 at location $(g, h)$; $(d_i, d_j)$ are degrees of the nodes $i$ and $j$ (stacked as elements on the diagonal matrix $D$ see Sec 2); and $P = \big[\frac{d_i^{-3/2}}{2}Z_{ii} + \frac{d_j^{-3/2}}{2}Z_{jj}\big]$ having non-zero entries only on the diagonal. Simplifying the expression further, we get

$$\vec{u}^T\frac{dL}{da_{ij}}\vec{u} = \vec{u}^T D^{-1/2}\big[Z_{ij} + Z_{ji}\big]D^{-1/2}\vec{u} - \vec{u}^T PD^{1/2}\big[D^{-1/2}AD^{-1/2}\big]\vec{u}$$
$$- \vec{u}^T\big[D^{-1/2}AD^{-1/2}\big]D^{1/2}P\vec{u}. \tag{8}$$

Using the fact that $D^{-1/2}AD^{-1/2}\vec{u} = L\vec{u} = \lambda\vec{u}$, and $PD^{1/2} = D^{1/2}P$ as both $P$ and $D$ are diagonal, the above equation reduces to:

$$\vec{u}^T\frac{dL}{da_{ij}}\vec{u} = \vec{u}^T D^{-1/2}\big[Z_{ij} + Z_{ji}\big]D^{-1/2}\vec{u} - 2\lambda\vec{u}^T PD^{1/2}\vec{u}$$
$$= \vec{u}^T D^{-1/2}\big[Z_{ij} + Z_{ji}\big]D^{-1/2}\vec{u} - \lambda\vec{u}^T\big[d_i^{-1}Z_{ii} + d_j^{-1}Z_{jj}\big]\vec{u}. \tag{9}$$

The scalar form of this expression is used in Eq.5.